# AN INFORMATION THEORETIC APPROACH TO RULE-BASED CONNECTIONIST EXPERT SYSTEMS

Rodney M. Goodman, John W. Miller
Department of Electrical Engineering
Caltech 116-81
Pasadena, CA 91125

Padhraic Smyth
Communication Systems Research
Jet Propulsion Laboratories 238-420
4800 Oak Grove Drive
Pasadena, CA 91109

## Abstract

We discuss in this paper architectures for executing probabilistic rule-bases in a parallel manner, using as a theoretical basis recently introduced information-theoretic models. We will begin by describing our (non-neural) learning algorithm and theory of quantitative rule modelling, followed by a discussion on the exact nature of two particular models. Finally we work through an example of our approach, going from database to rules to inference network, and compare the network's performance with the theoretical limits for specific problems.

## Introduction

With the advent of relatively cheap mass storage devices it is common in many domains to maintain large databases or logs of data, e.g., in telecommunications, medicine, finance, etc. The question naturally arises as to whether we can extract models from the data in an automated manner and use these models as the basis for an autonomous rational agent in the given domain, i.e., automatically generate "expert systems" from data. There are really two aspects to this problem: firstly *learning* a model and, secondly, performing *inference* using this model. What we propose in this paper is a rather novel and hybrid approach to learning and inference. Essentially we combine the qualitative knowledge representation ideas of AI with the distributed computational advantages of connectionist models, using an underlying theoretical basis tied to information theory. The knowledge representation formalism we adopt is the rule-based representation, a scheme which is well supported by cognitive scientists and AI researchers for modeling higher level symbolic reasoning tasks. We have recently developed an information-theoretic algorithm called ITRULE which extracts an optimal set of probabilistic rules from a given data set [1, 2, 3]. It must be emphasised that we do *not* use any form of neural learning such as backpropagation in our approach. To put it simply, the ITRULE learning algorithm is far more computationally direct and better understood than (say) backpropagation for this particular learning task of finding the most informative individual rules without reference to their collective properties. Performing useful inference with this model or set of rules, is quite a difficult problem. Exact theoretical schemes such as maximum entropy (ME) are intractable for real-time applications.

We have been investigating schemes where the rules represent links on a directed graph and the nodes correspond to propositions, i.e., variable-value pairs. Our approach is characterised by loosely connected, multiple path (arbitrary topology) graph structures, with nodes performing local non-linear decisions as to their true state based on both supporting evidence and their *a priori* bias. What we have in fact is a recurrent neural network. What is different about this approach compared to a standard connectionist model as learned by a weight-adaptation algorithm such as BP? The difference lies in the *semantics* of the representation [4]. Weights such as log-odds ratios based on log transformations of probabilities possess a clear meaning to the user, as indeed do the nodes themselves. This explicit representation of knowledge is a key requirement for any system which purports to perform reasoning, probabilistic or otherwise. Conversely, the lack of explicit knowledge representation in most current connectionist approaches, i.e., the "black box" syndrome, is a major limitation to their application in critical domains where user-confidence and explanation facilities are key criteria for deployment in the field.

## Learning the model

Consider that we have $M$ observations or samples available, e.g., the number of items in a database. Each sample datum is described in terms of $N$ attributes or features, which can assume values in a corresponding set of $N$ discrete alphabets. For example our data might be described in the form of 10-component binary vectors. The requirement for *discrete* rather than *continuous*-valued attributes is dictated by the very nature of the rule-based representation. In addition it is important to note that we do *not* assume that the sample data is somehow exhaustive and "correct." There is a tendency in both the neural network and AI learning literature to analyse learning in terms of learning a Boolean function from a truth table. The implicit assumption is often made that given enough samples, and a good enough learning algorithm we can always learn the function exactly. This is a fallacy, since it depends on the feature representation. For any problem of interest there are always hidden causes with a consequent non-zero Bayes misclassification risk, i.e., the function is dependent on non-observable features (unseen columns of the truth table). Only in artificial problems such as game playing is "perfect" classification possible — in practical problems nature hides the real features. This phenomenon is well known in the statistical pattern recognition literature and renders invalid those schemes which simply try to perfectly classify or memorise the training data.

We use the following simple model of a rule, i.e.,

If $\mathbf{Y} = y$ then $\mathbf{X} = x$ with probability $p$

where $\mathbf{X}$ and $\mathbf{Y}$ are two attributes (random variables) with "x" and "y" being values in their respective discrete alphabets. Given sample data as described earlier we pose the problem as follows: can we find the "best" rules from a given data set, say the $K$ best rules ? We will refer to this problem as that of *generalised rule induction*, in order to distinguish it from the special case of deriving classification

rules. Clearly we require both a preference measure to rank the rules and a learning algorithm which uses the preference measure to find the $K$ best rules.

Let us define the information which the *event y* yields about the variable **X**, say $f(\mathbf{X}; y)$. Based on the requirements that $f(\mathbf{X}; y)$ is both non-negative and that its expectation with respect to **Y** equals the average mutual information $I(\mathbf{X}; \mathbf{Y})$, Blachman [5] showed that the *only* such function is the j-measure, which is defined as

$$j(\mathbf{X}; y) \; = \; p(x|y) \log\left(\frac{p(x|y)}{p(x)}\right) + p(\bar{x}|y) \log\left(\frac{p(\bar{x}|y)}{p(\bar{x})}\right)$$

More recently we have shown that $j(\mathbf{X}; y)$ possesses unique properties as a rule information measure [6]. In general the j-measure is the average change in bits required to specify **X** between the *a priori* distribution ($p(\mathbf{X})$) and the *a posteriori* distribution ($p(\mathbf{X}|y)$). It can also be interpreted as a special case of the cross-entropy or binary discrimination (Kullback [7]) between these two distributions. We further define $J(\mathbf{X}; y)$ as the *average* information content where $J(\mathbf{X}; y) \; = \; p(y).j(\mathbf{X}; y)$. $J(\mathbf{X}; y)$ simply weights the instantaneous rule information $j(\mathbf{X}; y)$ by the probability that the left-hand side will occur, i.e., that the rule will be fired. This definition is motivated by considerations of learning useful rules in a resource-constrained environment. A rule with high information content must be both a good predictor and have a reasonable probability of being fired, i.e., $p(y)$ can not be too small. Interestingly enough our definition of $J(\mathbf{X}; y)$ possesses a well-defined interpretation in terms of classical induction theory, trading off hypothesis simplicity with the goodness-of-fit of the hypothesis to the data [8].

The ITRULE algorithm [1, 2, 3] uses the J-measure to derive the most informative set of rules from an input data set. The algorithm produces a set of $K$ probabilistic rules, ranked in order of decreasing information content. The parameter $K$ may be user-defined or determined via some statistical significance test based on the size of the sample data set available. The algorithm searches the space of possible rules, trading off generality of the rules with their predictiveness, and using information-theoretic bounds to constrain the search space.

## Using the Model to Perform Inference

Having learned the model we now have at our disposal a set of lower order constraints on the $N$-th order joint distribution in the form of probabilistic rules. This is our *a priori* model. In a typical inference situation we are given some initial conditions (i.e., some nodes are clamped), we are allowed to measure the state of some other nodes (possibly at a cost), and we wish to infer the state or probability of one more goal propositions or nodes from the available evidence. It is important to note that this is a much more difficult and general problem than classification of a single, fixed, goal variable, since both the initial conditions and goal propositions may vary considerably from one problem instance to the next. This is the inference problem, determining an *a posteriori* distribution in the face of incomplete and uncertain information. The exact maximum entropy solution to this problem is in-

tractable and, despite the elegance of the problem formulation, stochastic relaxation techniques (Geman [9]) are at present impractical for real-time robust applications. Our motivation then is to perform an approximation to exact Bayesian inference in a robust manner. With this in mind we have developed two particular models which we describe as the hypothesis testing network and the uncertainty network.

**Principles of the Hypothesis Testing Network**

In the first model under consideration each directed link from $y$ to $x$ is assigned a weight corresponding to the weight of evidence of $y$ on $x$. This idea is not necessarily new, although our interpretation and approach is different to previous work [10, 4]. Hence we have

$$W_{xy} = \log \frac{p(x|y)}{p(x)} - \log \frac{p(\overline{x}|y)}{p(\overline{x})} \quad \text{and} \quad R_x = -\log \frac{p(x)}{p(\overline{x})}$$

and the node $x$ is assigned a threshold term corresponding to *a priori* bias. We use a sigmoidal activation function, i.e.,

$$a(x) = \frac{1}{1 + e^{\frac{\Delta E_x - R_x}{T}}} \quad \text{where} \quad \Delta E_x = \sum_{i=1}^{n} W_{xy_i} \cdot a(y_i) - R_x$$

based on multiple binary inputs $y_1 ... y_n$ to $x$. Let $S$ be the set of all $y_i$ which are hypothesised true (i.e., $a(y_i) = 1$), so that

$$\Delta E_x = \log \frac{p(x)}{p(\overline{x})} + \sum_{y_i \in S} \left( \log \frac{p(x|y_i)}{p(x)} - \log \frac{p(\overline{x}|y_i)}{p(\overline{x})} \right)$$

If each $y_i$ is conditionally independent given $x$ then we can write

$$\frac{p(x|S)}{p(\overline{x}|S)} = \frac{p(x)}{p(\overline{x})} \prod_{y_i \in S} \frac{p(x|y_i)}{p(\overline{x}|y_i)}$$

Therefore the updating rule for conditionally independent $y_i$ is:

$$T \cdot \log \frac{a(x)}{1 - a(x)} = \log \frac{p(x|S)}{1 - p(x|S)}$$

Hence $a(x) > \frac{1}{2}$ iff $p(x|S) > \frac{1}{2}$ and if $T = 1$, a(x) is exactly $p(x|S)$. In terms of a hypothesis test, a(x) is chosen true iff:

$$\sum \log \frac{p(x|y_i)}{p(\overline{x}|y_i)} \geq -\log \frac{p(x)}{p(\overline{x})}$$

Since this describes the Neyman-Pearson decision region for independent measurements (evidence or $y_i$) with $R_x = -\log \frac{p(x)}{p(\overline{x})}$ [11], this model can be interpreted as a distributed form of hypothesis testing.

**Principles of the Uncertainty Network**

For this model we defined the weight on a directed link from $y_i$ to $x$ as

$$w_{xy_i} = s_i . j(X; y_i) = s_i . \left( p(x|y_i) \log\left(\frac{p(x|y_i)}{p(x)}\right) + p(\bar{x}|y_i) \log\left(\frac{p(\bar{x}|y_i)}{p(\bar{x})}\right) \right)$$

where $s_i = \pm 1$ and the threshold is the same as the hypothesis model. We can interpret $w_{xy_i}$ as the change in bits to specify the *a posteriori* distribution of $x$. If $p(x|y_i) \geq p(x)$, $w_{xy_i}$ has positive support for $x$, i.e., $s_i = +1$. If $p(x|y_i) < p(x)$, $w_{xy_i}$ has negative support for $x$, i.e., $s_i = -1$. If we interpret the activation $a(y_i)$ as an estimator $(\hat{p}(y))$ for $p(y_i)$, then for multiple inputs,

$$\Delta E_x = \sum_i a(y_i) . w_{xy_i} - R_x$$

$$= \sum_i \hat{p}(y_i) . s_i . \left( p(x|y_i) \log\left(\frac{p(x|y_i)}{p(x)}\right) + p(\bar{x}|y_i) \log\left(\frac{p(\bar{x}|y_i)}{p(\bar{x})}\right) \right)$$

This sum over input links weighted by activation functions can be interpreted as the total directional change in bits required to specify $x$, *as calculated locally by the node $x$*. One can normalise $\Delta E_x$ to obtain an average change in bits by dividing by a suitable temperature $T$. The node $x$ can make a local decision by recovering $\hat{p}(x)$ from an inverse J-measure transformation of $\Delta E$ (the sigmoid is an approximation to this inverse function).

## Experimental Results and Conclusions

In this section we show how rules can be generated from example data and automatically incorporated into a parallel inference network that takes the form of a multi-layer neural network. The network can then be "run" to perform parallel inference. The domain we consider is that of a financial database of mutual funds, using published statistical data [12]. The approach is, however, typical of many different real world domains.

Figure 1 shows a portion of a set of typical raw data on no-load mutual funds. Each line is an instance of a fund (with name omitted), and each column represents an attribute (or feature) of the fund. Attributes can be numerical or categorical. Typical categorical attributes are the fund type which reflect the investment objectives of the fund (growth, growth and income, balanced, and agressive growth) and a typical numerical attribute is the five year return on investment expressed as a percentage. There are a total of 88 fund examples in this data set. From this raw data a second quantized set of the 88 examples is produced to serve as the input to ITRULE (Figure 2). In this example the attributes have been categorised to binary values so that they can be directly implemented as binary neurons. The ITRULE software then processes this table to produce a set of rules. The rules are ranked in order of decreasing information according to the J-measure. Figure 3 shows a

portion (the top ten rules) of the ITRULE output for the mutual fund data set. The hypothesis test log-likelihood metric $h(X;y)$, the instantaneous j-measure $j(X;y)$, and the average J-measure $J(X;y)$, are all shown, together with the rule transition probability $p(x/y)$.

In order to perform inference with the ITRULE rules we need to map the rules into a neural inference net. This is automatically done by ITRULE which generates a network file that can be loaded into a neural network simulator. Thus rule information metrics become connection weights. Figure 4 shows a typical network derived from the ITRULE rule output for the mutual funds data. For clarity not all the connections are shown. The architecture consists of two layers of neurons (or "units"): an input layer and an output layer, both of which have an activation within the range $\{0, 1\}$. There is one unit in the input layer (and a corresponding unit in the output layer) for each attribute in the mutual funds data. The output feeds back to the input layer, and each layer is synchronously updated. The output units can be considered to be the right hand sides of the rules and thus receive inputs from many rules, where the strength of the connection is the rule's metric. The output units implement a sigmoid activation function on the sum of the inputs, and thus compute an activation which is an estimator of the right hand side posteriori attribute value. The input units simply pass this value on to the output layer and thus have a linear activation.

To perform inference on the network, a probe vector of attribute values is loaded into the input and output layers. Known values are clamped and cannot change while unknown or desired attribute values are free to change. The network then relaxes and after several feedback cycles converges to a solution which can be read off the input or output units. To evaluate the models we setup four standard classification tests with varying number of nodes clamped as inputs. Unclamped nodes were set to their *a priori* probability. After relaxing the network, the activation of the "target" node was compared with the true attribute values for that sample in order to determine classification performance. The two models were each trained on 10 randomly selected sets of 44 samples. The performance results given in Table 1 are the average classification rate of the models on the other 44 *unseen* samples. The Bayes risk (for a uniform loss matrix) of each classification test was calculated from the 88 samples. The actual performance of the networks occasionally exceeded this value due to small sample variations on the 44/44 cross validations.

Table 1

| Units Clamped | Uncertainty Test | Hypothesis Test | 1 - Bayes' Risk |
|:---:|:---:|:---:|:---:|
| 9 | 66.8% | 70.4% | 88.6% |
| 5 | 70.1% | 70.1% | 80.6% |
| 2 | 48.2% | 63.0% | 63.6% |
| 1 | 51.4% | 65.7% | 64.8% |

We conclude from the performance of the networks as classifiers that they have indeed learned a model of the data using a rule-based representation. The hypothesis network performs slightly better than the uncertainty model, with both being quite close to the estimated optimal rate (the Bayes' risk). Given that we know that the independence assumptions in both models do not hold exactly, we coin the term *robust inference* to describe this kind of accurate behaviour in the presence of incomplete and uncertain information. Based on these encouraging initial results, our current research is focusing on higher-order rule networks and extending our theoretical understanding of models of this nature.

## Acknowledgments

This work is supported in part by a grant from Pacific Bell, and by Caltech's program in Advanced Technologies sponsored by Aerojet General, General Motors and TRW. Part of the research described in this paper was carried out by the Jet Propulsion Laboratory, California Institute of Technology, under a contract with the National Aeronautics and Space Administration. John Miller is supported by NSF grant no. ENG-8711673.

## References

1. R. M. Goodman and P. Smyth, 'An information theoretic model for rule-based expert systems,' presented at the 1988 International Symposium on Information Theory, Kobe, Japan.
2. R. M. Goodman and P. Smyth, 'Information theoretic rule induction,' *Proceedings of the 1988 European Conference on AI*, Pitman Publishing: London.
3. R. M. Goodman and P. Smyth, 'Deriving rules from databases: the ITRULE algorithm,' submitted for publication.
4. H. Geffner and J. Pearl, 'On the probabilistic semantics of connectionist networks,' *Proceedings of the 1987 IEEE ICNN*, vol. II, pp. 187–195.
5. N. M. Blachman, 'The amount of information that y gives about X,' *IEEE Transactions on Information Theory*, vol. IT–14 (1), 27–31, 1968.
6. P. Smyth and R. M. Goodman, 'The information content of a probabilistic rule,' submitted for publication.
7. S. Kullback, *Information Theory and Statistics*, New York: Wiley, 1959.
8. D. Angluin and C. Smith, 'Inductive inference: theory and methods,' *ACM Computing Surveys, 15(3)*, pp. 237-270, 1984.
9. S. Geman, 'Stochastic relaxation methods for image restoration and expert systems,' in *Maximum Entropy and Bayesian Methods in Science and Engineering (Vol. 2)*, 265–311, Kluwer Academic Publishers, 1988.
10. G. Hinton and T. Sejnowski, 'Optimal perceptual inference,' *Proceedings of the IEEE CVPR 1983*.
11. R. E. Blahut, *Principles and Practice of Information Theory*, Addison-Wesley: Reading, MA, 1987.
12. American Association of Investors, *The individual investor's guide to no-load mutual funds*, International Publishing Corporation: Chicago, 1987.

| Fund Type | 5 Year Return % | Diver-sity | Beta (Risk) | Bull Perf. | Bear Perf. | Stocks % | Invest-ment Incm. $ | Net Asset Value $ | Distri-butions (%NAV) | Expense Ratio % | Turn-over Rate % | Total Assets $M |
|---|---|---|---|---|---|---|---|---|---|---|---|---|
| Balanced | 136 | C | 0.8 | B | D | 87 | 0.67 | 37.3 | 17.63 | 0.79 | 34 | 415 |
| Growth | 32.5 | C | 1.05 | E | B | 81 | -0.02 | 12.5 | 0.88 | 1.4 | 200 | 16 |
| Growth&Income | 88.3 | A | 0.96 | C | D | 82 | 0.14 | 11.9 | 4.78 | 1.34 | 127 | 27 |
| Agressive | -24 | A | 1.23 | E | E | 95 | 0.02 | 6.45 | 9.30 | 1.4 | 161 | 64 |
| Growth&Income | 172 | E | 0.59 | A | B | 73 | 0.53 | 13.6 | 9.97 | 1.09 | 31 | 113 |
| Balanced | 144 | C | 0.71 | B | B | 51 | 0.72 | 13 | 10.44 | 0.98 | 239 | 190 |

**Figure 1.**    Raw Mutual Funds Data

| Type A | Type B | Type G | Type GI | 5 Year Return % | Beta | Stocks >90% | Turn-over | Assets | Distri-butions | Diver-sity | Bull Perf. | Bear Perf. |
|---|---|---|---|---|---|---|---|---|---|---|---|---|
| | | | | S&P=138% above S&P below S&P | | | <100% >100% | <$100M >$100M | <15%NAV >15%NAV | C,D,E A,B | C,D,E A,B | C,D,E A,B |
| no | no | yes | no | below | under1 | no | low | large | high | low | high | low |
| no | no | yes | no | below | over1 | no | high | small | low | low | low | high |
| no | no | no | yes | below | under1 | no | high | small | low | high | low | low |
| no | no | no | yes | above | under1 | no | low | large | low | low | high | high |
| no | no | no | yes | below | under1 | yes | low | small | high | high | low | high |
| no | no | yes | no | above | under1 | no | low | large | high | high | high | low |

**Figure 2.**    Quantized Mutual Funds Data

| ITRULE rule output: Mutual Funds | | | | | | p(x/y) | j(X;y) | J(X;y) | h(X;y) |
|---|---|---|---|---|---|---|---|---|---|
| 1 IF | 5yrRet>S&P | above | THEN | Bull_perf | high | 0.97 | 0.75 | 0.235 | 4.74 |
| 2 IF | Bull_perf | low | THEN | 5yrRet>S&P | below | 0.98 | 0.41 | 0.201 | 4.31 |
| 3 IF | Assets | large | THEN | Bull_perf | high | 0.81 | 0.28 | 0.127 | 2.02 |
| 4 IF | Bull_perf | high | THEN | 5yrRet>S&P | above | 0.40 | 0.25 | 0.127 | -1.71 |
| 5 IF | typeA | yes | THEN | typeG | no | 0.04 | 0.50 | 0.123 | -3.87 |
| 6 IF | Bull_perf | low | THEN | Assets | small | 0.18 | 0.25 | 0.121 | -1.95 |
| 7 IF | typeGI | yes | THEN | typeG | no | 0.05 | 0.49 | 0.109 | -3.74 |
| 8 IF | Bull_perf | high | THEN | Assets | large | 0.72 | 0.21 | 0.109 | 1.64 |
| 9 IF | typeG | yes | THEN | typeA | no | 0.97 | 0.27 | 0.108 | 3.54 |
| 10 IF | Assets | small | THEN | Bull_perf | low | 0.26 | 0.19 | 0.103 | -1.57 |

**Figure 3.** Top Ten Mutual Funds Rules

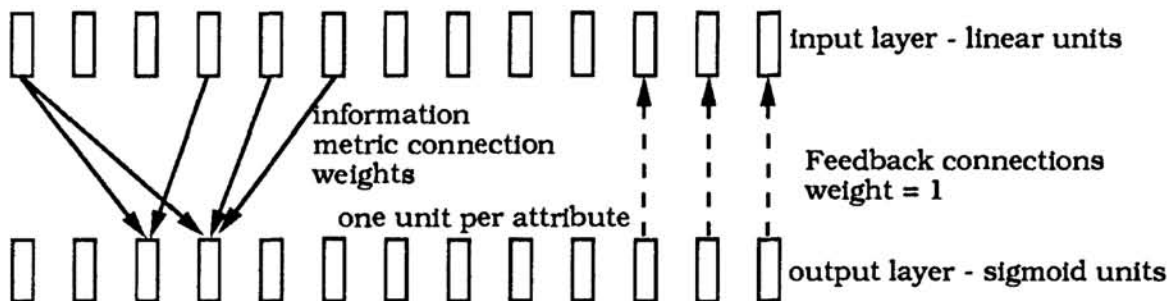

**Figure 4.**  Rule Network